# Correlations strike back (again): the case of associative memory retrieval

**Cristina Savin**[1]
cs664@cam.ac.uk

**Peter Dayan**[2]
dayan@gatsby.ucl.ac.uk

**Máté Lengyel**[1]
m.lengyel@eng.cam.ac.uk

[1]Computational & Biological Learning Lab, Dept. Engineering, University of Cambridge, UK
[2]Gatsby Computational Neuroscience Unit, University College London, UK

## Abstract

It has long been recognised that statistical dependencies in neuronal activity need to be taken into account when decoding stimuli encoded in a neural population. Less studied, though equally pernicious, is the need to take account of dependencies between synaptic weights when decoding patterns previously encoded in an auto-associative memory. We show that activity-dependent learning generically produces such correlations, and failing to take them into account in the dynamics of memory retrieval leads to catastrophically poor recall. We derive optimal network dynamics for recall in the face of synaptic correlations caused by a range of synaptic plasticity rules. These dynamics involve well-studied circuit motifs, such as forms of feedback inhibition and experimentally observed dendritic nonlinearities. We therefore show how addressing the problem of synaptic correlations leads to a novel functional account of key biophysical features of the neural substrate.

## 1   Introduction

Auto-associative memories have a venerable history in computational neuroscience. However, it is only rather recently that the statistical revolution in the wider field has provided theoretical traction for this problem [1]. The idea is to see memory storage as a form of lossy compression – information on the item being stored is mapped into a set of synaptic changes – with the neural dynamics during retrieval representing a biological analog of a corresponding decompression algorithm. This implies there should be a tight, and indeed testable, link between the learning rule used for encoding and the neural dynamics used for retrieval [2].

One issue that has been either ignored or trivialized in these treatments of recall is correlations among the synapses [1–4] – beyond the perfect (anti-)correlations emerging between reciprocal synapses with precisely (anti-)symmetric learning rules [5]. There is ample experimental data for the existence of such correlations: for example, in rat visual cortex, synaptic connections tend to cluster together in the form of overrepresented patterns, or motifs, with reciprocal connections being much more common than expected by chance, and the strengths of the connections to and from each neuron being correlated [6]. The study of neural coding has indicated that it is essential to treat correlations in neural activity appropriately in order to extract stimulus information well [7–9]. Similarly, it becomes pressing to examine the nature of correlations among synaptic weights in auto-associative memories, the consequences for retrieval of ignoring them, and methods by which they might be accommodated.

Here, we consider several well-known learning rules, from simple additive ones to bounded synapses with metaplasticity, and show that, with a few significant exceptions, they induce correlations between synapses that share a pre- or a post-synaptic partner. To assess the importance of these dependencies for recall, we adopt the strategy of comparing the performance of decoders which either do or do not take them into account [10], showing that they do indeed have an important effect on efficient retrieval. Finally, we show that approximately optimal retrieval involves particular forms of nonlinear interactions between different neuronal inputs, as observed experimentally [11].

## 2   General problem formulation

We consider a network of $N$ binary neurons that enjoy all-to-all connectivity.[1] As is conventional, and indeed plausibly underpinned by neuromodulatory interactions [12], we assume that network dynamics do not play a role during storage (with stimuli being imposed as patterns of activity on the neurons), and that learning does not occur during retrieval.

To isolate the effects of different plasticity rules on synaptic correlations from other sources of correlations, we assume that the patterns of activity inducing the synaptic changes have no particular structure, i.e. their distribution factorizes. For further simplicity, we take these activity patterns to be binary with pattern density $f$, i.e. a prior over patterns defined as:

$$\mathrm{P_{store}}(\mathbf{x}) \quad = \quad \prod_i \mathrm{P_{store}}(x_i) \qquad \mathrm{P_{store}}(x_i) = f^{x_i} \cdot (1-f)^{1-x_i} \qquad (1)$$

During recall, the network is presented with a cue, $\tilde{\mathbf{x}}$, which is a noisy or partial version of one of the originally stored patterns. Network dynamics should complete this partial pattern, using the information in the weights $\mathbf{W}$ (and the cue). We start by considering arbitrary dynamics; later we impose the critical constraint for biological realisability that they be strictly local, i.e. the activity of neuron $i$ should depend exclusively on inputs through incoming synapses $\mathbf{W}_{i,\cdot}$.

Since information storage by synaptic plasticity is lossy, recall is inherently a probabilistic inference problem [1, 13] (Fig. 1a), requiring estimation of the posterior over patterns, given the information in the weights and the recall cue:

$$\mathrm{P}\left(\mathbf{x}|\mathbf{W}, \tilde{\mathbf{x}}\right) \propto \mathrm{P_{store}}(\mathbf{x}) \cdot \mathrm{P_{noise}}(\tilde{\mathbf{x}}|\mathbf{x}) \cdot \mathrm{P}(\mathbf{W}|\mathbf{x}) \qquad (2)$$

This formulation has formed the foundation of recent work on constructing efficient autoassociative recall dynamics for a range of different learning rules [2–4]. In this paper, we focus on the last term $\mathrm{P}(\mathbf{W}|\mathbf{x})$, which expresses the probability of obtaining $\mathbf{W}$ as the synaptic weight matrix when $\mathbf{x}$ is stored along with $T-1$ random patterns (sampled from the prior, Eq. 1). Critically, this is where we diverge from previous analyses that assumed this distribution was factorised, or only trivially correlated due to reciprocal synapses being precisely (anti-)symmetric [1, 2, 4]. In contrast, we explicitly study the emergence and effects of non-trivial correlations in the synaptic weight matrix-distribtion, because almost all synaptic plasticity rules induce statistical dependencies between the synaptic weights of each neuron (Fig. 1a, d).

The inference problem expressed by Eq. 2 can be translated into neural dynamics in several ways – dynamics could be deterministic, attractor-like, converging to the most likely pattern (a MAP estimate) of the distribution of $\mathbf{x}$ [2], or to a mean-field approximate solution [3]; alternatively, the dynamics could be stochastic, with the activity over time representing samples from the posterior, and hence implicitly capturing the uncertainty associated with the answer [4]. We consider the latter. Since we estimate performance by average errors, the optimal response is the mean of the posterior, which can be estimated by integrating the activity of the network during retrieval.

We start by analysing the class of additive learning rules, to get a sense for the effect of correlations on retrieval. Later, we focus on multi-state synapses, for which learning rules are described by transition probabilities between the states [14]. These have been used to capture a variety of important biological constraints such as bounds on synaptic strengths and metaplasticity, i.e. the fact that synaptic changes induced by a certain activity pattern depend on the history of activity at the synapse [15]. The two classes of learning rule are radically different; so if synaptic correlations matter during retrieval in both cases, then the conclusion likely applies in general.

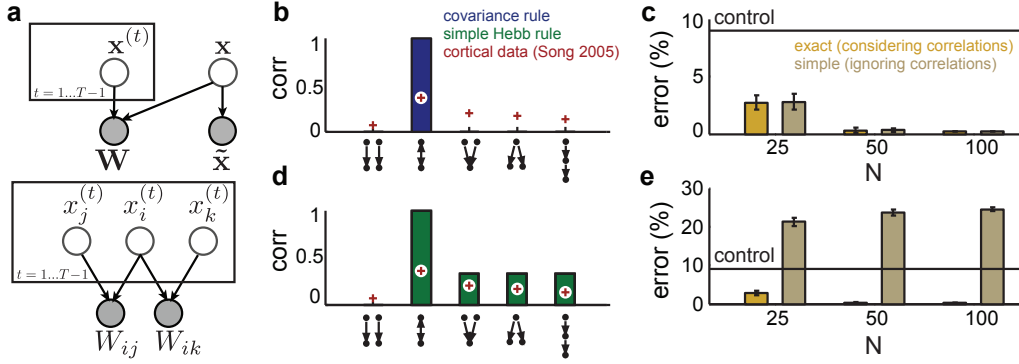

Figure 1: **Memory recall as inference and additive learning rules. a.** Top: Synaptic weights, $\mathbf{W}$, arise by storing the target pattern $\mathbf{x}$ together with $T-1$ other patterns, $\{\mathbf{x}^{(t)}\}_{t=1\ldots T-1}$. During recall, the cue, $\tilde{\mathbf{x}}$, is a noisy version of the target pattern. The task of recall is to infer $\mathbf{x}$ given $\mathbf{W}$ and $\tilde{\mathbf{x}}$ (by marginalising out $\{\mathbf{x}^{(t)}\}$). Bottom: The activity of neuron $i$ across the stored patterns is a source of shared variability between synapses connecting it to neurons $j$ and $k$. **b-c.** Covariance rule: patterns of synaptic correlations and recall performance for retrieval dynamics ignoring or considering synaptic correlations; $T = 5$. **d-e.** Same for the simple Hebbian learning rule. The control is an optimal decoder that ignores $\mathbf{W}$.

## 3 Additive learning rules

Local additive learning rules assume that synaptic changes induced by different activity patterns combine additively; such that storing a sequence of $T$ patterns from $\mathrm{P}_{\mathrm{store}}(\mathbf{x})$, results in weights $W_{ij} = \sum_t \Omega(x_i^{(t)}, x_j^{(t)})$, with function $\Omega(x_i, x_j)$ describing the change in synaptic strength induced by presynaptic activity $x_j$ and postsynaptic activity $x_i$. We consider a generalized Hebbian form for this function, with $\Omega(x_i, x_j) = (x_i - \alpha)(x_j - \beta)$. This class includes, for example, the covariance rule ($\alpha = \beta = f$), classically used in Hopfield networks, or simple Hebbian learning ($\alpha = \beta = 0$).

As synaptic changes are deterministic, the only source of uncertainty in the distribution $\mathrm{P}(\mathbf{W}|\mathbf{x})$ is the identity of the other stored patterns. To estimate this, let us first consider the distribution of the weights after storing one random pattern from $\mathrm{P}_{\mathrm{store}}(\mathbf{x})$. The mean $\boldsymbol{\mu}$ and covariance $\mathbf{C}$ of the weight change induced by this event can be computed as:[2]

$$\boldsymbol{\mu} = \int \mathrm{P}_{\mathrm{store}}(\mathbf{x})\boldsymbol{\Omega}_|(\mathbf{x})d\mathbf{x}, \qquad \mathbf{C} = \int \mathrm{P}_{\mathrm{store}}(\mathbf{x})\left(\boldsymbol{\Omega}_|(\mathbf{x}) \cdot \boldsymbol{\Omega}_|(\mathbf{x})^{\mathrm{T}}\right)d\mathbf{x} - \boldsymbol{\mu}\cdot\boldsymbol{\mu}^{\mathrm{T}} \qquad (3)$$

Since the rule is additive and the patterns are independent, the mean and covariance scale linearly with the number of intervening patterns. Hence, the distribution over possible weight values at recall, given that pattern $\mathbf{x}$ is stored along with $T - 1$ other, random, patterns has mean $\boldsymbol{\mu}_W = \boldsymbol{\Omega}(\mathbf{x}) + (T - 1) \cdot \boldsymbol{\mu}$, and covariance $\mathbf{C}_W = (T - 1) \cdot \mathbf{C}$. Most importantly, because the rule is additive, in the limit of many stored patterns (and in practice even for modest values of $T$), the distribution $\mathrm{P}(\mathbf{W}|\mathbf{x})$ approaches a multivariate Gaussian that is characterized completely by these two quantities; moreover, its covariance is independent of $\mathbf{x}$.

For retrieval dynamics based on Gibbs sampling, the key quantity is the log-odds ratio

$$I_i = \log\left(\frac{\mathrm{P}(x_i = 1|\mathbf{x}_{\neg i}, \mathbf{W}, \tilde{\mathbf{x}})}{\mathrm{P}(x_i = 0|\mathbf{x}_{\neg i}, \mathbf{W}, \tilde{\mathbf{x}})}\right) \qquad (4)$$

for neuron $i$, which could be represented by the total current entering the unit. This would translate into a probability of firing given by the sigmoid activation function $f(I_i) = 1/(1 + e^{-I_i})$.

The total current entering a neuron is a sum of two terms: one term from the external input of the form $c_1 \cdot \tilde{x}_i + c_2$ (with constants $c_1$ and $c_2$ determined by parameters $f$ and $r$ [16]), and one term from the recurrent input, of the form:

$$I_i^{\mathrm{rec}} = \frac{1}{2(T-1)}\left(\left(\mathbf{W}_| - \boldsymbol{\mu}_W^{(0)}\right)^{\mathrm{T}}\mathbf{C}^{-1}\left(\mathbf{W}_| - \boldsymbol{\mu}_W^{(0)}\right) - \left(\mathbf{W}_| - \boldsymbol{\mu}_W^{(1)}\right)^{\mathrm{T}}\mathbf{C}^{-1}\left(\mathbf{W}_| - \boldsymbol{\mu}_W^{(1)}\right)\right) \qquad (5)$$

where $\boldsymbol{\mu}_W^{(0/1)} = \boldsymbol{\Omega}_{|}(\mathbf{x}^{(0/1)}) + (T{-}1)\boldsymbol{\mu}$ and $\mathbf{x}^{(0/1)}$ is the vector of activities obtained from $\mathbf{x}$ in which the activity of neuron $i$ is set to 0, or 1, respectively.

It is easy to see that for the covariance rule, $\Omega\left(x_i, x_j\right) = (x_i - f)(x_j - f)$, synapses sharing a single pre- or post-synaptic partner happen to be uncorrelated (Fig. 1b). Moreover, as for any (anti-)symmetric additive learning rule, reciprocal connections are perfectly correlated ($W_{ij} = W_{ji}$). The (non-degenerate part of the) covariance matrix in this case becomes diagonal, and the total current in optimal retrieval reduces to simple linear dynamics :

$$I_i = \frac{1}{(T-1)\,\sigma_W^2}\left(\underbrace{\sum_j W_{ij}x_j}_{\text{recurrent input}} - \underbrace{\frac{(1-2f)^2}{2}\sum_j x_j}_{\text{feedback inhibition}} - \underbrace{f\sum_j W_{ij}}_{\text{homeostatic term}} - \underbrace{f^2\frac{1-2f}{2}}_{\text{constant}}\right) \qquad (6)$$

where $\sigma_W^2$ is the variance of a synaptic weight resulting from storing a single pattern. This term includes a contribution from recurrent excitatory input, dynamic feedback inhibition (proportional to the total population activity) and a homeostatic term that reduces neuronal excitability as function of the net strength of its synapses (a proxy for average current the neuron expects to receive) [17]. Reassuringly, the optimal decoder for the covariance rule recovers a form for the input current that is closely related to classic Hopfield-like [5] dynamics (with external field [1, 18]): feedback inhibition is needed only when the stored patterns are not balanced ($f \neq 0.5$); for the balanced case, the homeostatic term can be integrated in the recurrent current, by rewriting neural activities as spins. In sum, for the covariance rule, synapses are fortuitously uncorrelated (except for symmetric pairs which are perfectly correlated), and thus simple, classical linear recall dynamics suffice (Fig. 1c).

The covariance rule is, however, the exception rather than the rule. For example, for simple Hebbian learning, $\Omega\left(x_i, x_j\right) = x_i \cdot x_j$, synapses sharing a pre- or post-synaptic partner are correlated (Fig. 1d) and so the covariance matrix $\mathbf{C}$ is no longer diagonal. Interestingly, the final expression of the recurrent current to a neuron remains strictly local (because of additivity and symmetry), and very similar to Eq. 6, but feedback inhibition becomes a *non-linear* function of the total activity in the network [16]. In this case, synaptic correlations have a dramatic effect: using the optimal non-linear dynamics ensures high performance, but trying to retrieve information using a decoder that assumes synaptic independence (and thus uses linear dynamics) yields extremely poor performance, which is even worse than the obvious control of relying only on the information in the recall cue and the prior over patterns (Fig. 1e).

For the generalized Hebbian case, $\Omega\left(x_i, x_j\right) = (x_i - \alpha)(x_j - \beta)$ with $\alpha \neq \beta$, the optimal decoder becomes even more complex, with the total current including additional terms accounting for pairwise correlations between any two synapses that have neuron $i$ as a pre- or post-synaptic partner [16]. Hence, retrieval is no longer strictly local[3] and a biological implementation will require approximating the contribution of non-local terms as a function of locally available information, as we discuss in detail for palimpsest learning below.

## 4 Palimpsest learning rules

Though additive learning rules are attractive for their analytical tractability, they ignore several important aspects of synaptic plasticity, e.g. they assume that synapses can grow without bound. We investigate the effects of bounded weights by considering another class of learning rules, which assumes synaptic efficacies can only take binary values, with stochastic transitions between the two underpinned by paired cascades of latent internal states [14] (Fig. 2). These learning rules, though very simple, capture an important aspect of memory – the fact that memory is leaky, and information about the past is overwritten by newly stored items (usually referred to as the palimpsest property). Additionally, such rules can account for experimentally observed synaptic metaplasticity [15].

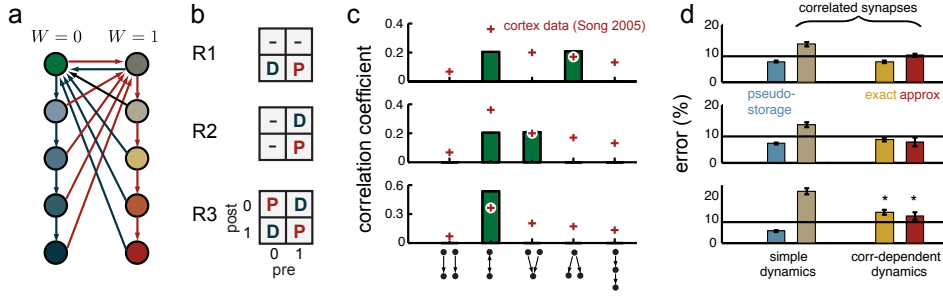

Figure 2: **Palimpsest learning. a.** The cascade model. Colored circles are latent states ($V$) that belong to two different synaptic weights ($W$), arrows are state transitions (blue: depression, red: potentiation) **b.** Different variants of mapping pre- and post-synaptic activations to depression (D) and potentiation (P): R1–postsynaptically gated, R2–presynaptically gated, R3–XOR rule. **c.** Correlation structure induced by these learning rules. **c.** Retrieval performance for each rule.

### Learning rule

Learning is stochastic and local, with changes in the state of a synapse $V_{ij}$ being determined only by the activation of the pre- and post-synaptic neurons, $x_j$ and $x_i$. In general, one could define separate transition matrices for each activity pattern, $\mathbf{M}(x_i, x_j)$, describing the probability of a synaptic state transitioning between any two states $V_{ij}$ to $V'_{ij}$ following an activity pattern, $(x_i, x_j)$. For simplicity, we define only two such matrices, for potentiation, $\mathbf{M}_+$, and depression, $\mathbf{M}_-$, respectively, and then map different activity patterns to these events. In particular, we assume Fusi's cascade model [14][4] and three possible mappings (Fig. 2b [16]): 1) a postsynaptically gated learning rule, where changes occur only when the postsynaptic neuron is active, with co-activation of pre- and post- neuron leading to potentiation, and to depression otherwise[5]; 2) a presynaptically gated learning rule, typically assumed when analysing cascades[20, 21]; and 3) an XOR-like learning rule which assumes potentiation occurs whenever the pre- and post- synaptic activity levels are the same, with depression otherwise. The last rule, proposed by Ref. 22, was specifically designed to eliminate correlations between synapses, and can be viewed as a version of the classic covariance rule fashioned for binary synapses.

### Estimating the mean and covariance of synaptic weights

At the level of a single synapse, the presentation of a sequence of uncorrelated patterns from $\mathrm{P}_{\text{store}}(x)$ corresponds to a Markov random walk, defined by a transition matrix $\overline{\mathbf{M}}$, which averages over possible neural activity patterns: $\overline{\mathbf{M}} = \sum_{x_i, x_j} \mathrm{P}_{\text{store}}(x_i) \cdot \mathrm{P}_{\text{store}}(x_j) \cdot \mathbf{M}(x_i, x_j)$. The distribution over synaptic states $t$ steps after the initial encoding can be calculated by starting from the stationary distribution of the weights $\boldsymbol{\pi}^{V0}$ (assuming a large number of other patterns have previously been stored; formally, this is the eigenvector of $\overline{\mathbf{M}}$ corresponding to eigenvalue $\lambda = 1$), then storing the pattern $(x_i, x_j)$, and finally $t - 1$ other patterns from the prior:

$$\boldsymbol{\pi}^V(x_i, x_j, t) = \overline{\mathbf{M}}^{t-1} \cdot \mathbf{M}(x_i, x_j) \cdot \boldsymbol{\pi}^{V0}, \tag{7}$$

with the distribution over states given as a column vector, $\pi_l^V = \mathrm{P}(V_{ij} = l | x_i, x_j)$, $l \in \{1 \dots 2n\}$, where $n$ is the depth of the cascade. Lastly, the distribution over weights, $\mathrm{P}(W_{ij} | x_i, x_j)$, can be derived as $\boldsymbol{\pi}^W = \mathbf{M}_{V \to W} \cdot \boldsymbol{\pi}^V$, where $\mathbf{M}_{V \to W}$ is a deterministic map from states to observed weights (Fig. 2**a**).

As in the additive case, the states of synapses sharing a pre- or post- synaptic partner will be correlated (Figs. 1a, 2c). The degree of correlations for different synaptic configurations can be estimated by generalising the above procedure to computing the joint distribution of the states of pairs of synapses, which we represent as a matrix $\boldsymbol{\rho}$. E.g. for a pair of synapses sharing a postsynaptic partner (Figs. 1b, d, and 2c), element $(u, v)$ is $\rho_{uv} = \mathrm{P}(V_{\text{post,pre1}} = u, V_{\text{post,pre2}} = v)$. Hence, the presentation of an activity pattern $(x_{\text{pre1}}, x_{\text{pre2}}, x_{\text{post}})$ induces changes in the corresponding pair of

incoming synapses to neuron post as $\boldsymbol{\rho}^{(1)} = \mathbf{M}(x_{\text{post}}, x_{\text{pre1}}) \cdot \boldsymbol{\rho}^{(0)} \cdot \mathbf{M}(x_{\text{post}}, x_{\text{pre2}})^{\text{T}}$, where $\boldsymbol{\rho}^{(0)}$ is the stationary distribution corresponding to storing an infinite number of triplets from the pattern distribution [16].

Replacing $\boldsymbol{\pi}^V$ with $\boldsymbol{\rho}$ (which is now a function of the triplet $(x_{\text{pre1}}, x_{\text{pre2}}, x_{\text{post}})$), and the multiplication by $\mathbf{M}$ with the slightly more complicated operator above, we can estimate the evolution of the joint distribution over synaptic states in a manner very similar to Eq. 7:

$$\boldsymbol{\rho}^{(t)} = \sum_{x_i} P_{\text{store}}(x_i) \cdot \hat{\mathbf{M}}(x_i) \cdot \boldsymbol{\rho}^{(t-1)} \cdot \hat{\mathbf{M}}(x_i)^{\text{T}}, \tag{8}$$

where $\hat{\mathbf{M}}(x_i) = \sum_{x_j} P_{\text{store}}(x_j) \mathbf{M}(x_i, x_j)$. Also as above, the final joint distribution over states can be mapped into a joint distribution over synaptic weights as $\mathbf{M}_{V \to W} \cdot \boldsymbol{\rho}^{(t)} \cdot \mathbf{M}_{V \to W}^{\text{T}}$. This approach can be naturally extended to all other correlated pairs of synapses [16].

The structure of correlations for different synaptic pairs varies significantly as a function of the learning rule (Fig. 2c), with the overall degree of correlations depending on a range of factors. Correlations tend to decrease with cascade depth and pattern sparsity. The first two variants of the learning rule considered are not symmetric, and so induce different patterns of correlations than the additive rules above. The XOR rule is similar to the covariance rule, but the reciprocal connections are no longer perfectly correlated (due to metaplasticity), which means that it is no longer possible to factorize $P(\mathbf{W}|\mathbf{x})$. Hence, assuming independence at decoding seems bound to introduce errors.

**Approximately optimal retrieval when synapses are independent**

If we ignore synaptic correlations, the evidence from the weights factorizes, $P(\mathbf{W}|\mathbf{x}) = \prod_{i,j} P(W_{ij}|x_i, x_j)$, and so the exact dynamics would be semi-local[3]. We can further approximate the contribution of the outgoing weights by its mean, which recovers the same simple dynamics derived for the additive case:

$$I_i = \log \left( \frac{P(x_i = 1 | \mathbf{x}_{\neg i}, \mathbf{W}, \tilde{\mathbf{x}})}{P(x_i = 0 | \mathbf{x}_{\neg i}, \mathbf{W}, \tilde{\mathbf{x}})} \right) = c_1 \sum_j W_{ij} x_j + c_2 \sum_j W_{ij} + c_3 \sum_j x_j + c_4 \tilde{x}_i + c_5 \tag{9}$$

The parameters $c$ depend on the prior over $x$, the noise model, the parameters of the learning rule and $t$. Again, the optimal decoder is similar to previously derived attractor dynamics; in particular, for stochastic binary synapses with presynaptically gated learning the optimal dynamics require dynamic inhibition only for sparse patterns, and no homeostatic term, as used in [21].

To validate these dynamics, we remove synaptic correlations by a pseudo-storage procedure in which synapses are allowed to evolve independently according to transition matrix $\overline{\mathbf{M}}$, rather than changing as actual intermediate patterns are stored. The dynamics work well in this case, as expected (Fig. 2d, blue bars). However, when storing actual patterns drawn from the prior, performance becomes extremely poor, and often worse than the control (Fig. 2d, gray bars). Moreover, performance worsens as the network size increases (not shown). Hence, ignoring correlations is highly detrimental for this class of learning rules too.

**Approximately optimal retrieval when synapses are correlated**

To accommodate synaptic correlations, we approximate $P(\mathbf{W}|\mathbf{x})$ with a maximum entropy distribution with the same marginals and covariance structure, ignoring the higher order moments.[6] Specifically, we assume the evidence from the weights has the functional form:

$$P(\mathbf{W}|\mathbf{x}, t) = \frac{1}{Z(\mathbf{x}, t)} \exp \left( \sum_{ij} k_{ij}(\mathbf{x}, t) \cdot W_{ij} + \frac{1}{2} \sum_{ijkl} J_{(ij)(kl)}(\mathbf{x}, t) \cdot W_{ij} W_{kl} \right) \tag{10}$$

We use the TAP mean-field method [23] to find parameters $\mathbf{k}$ and $\mathbf{J}$ and the partition function, $Z$, for each possible activity pattern $\mathbf{x}$, given the mean and covariance for the synaptic weights matrix, computed above[7] [16].

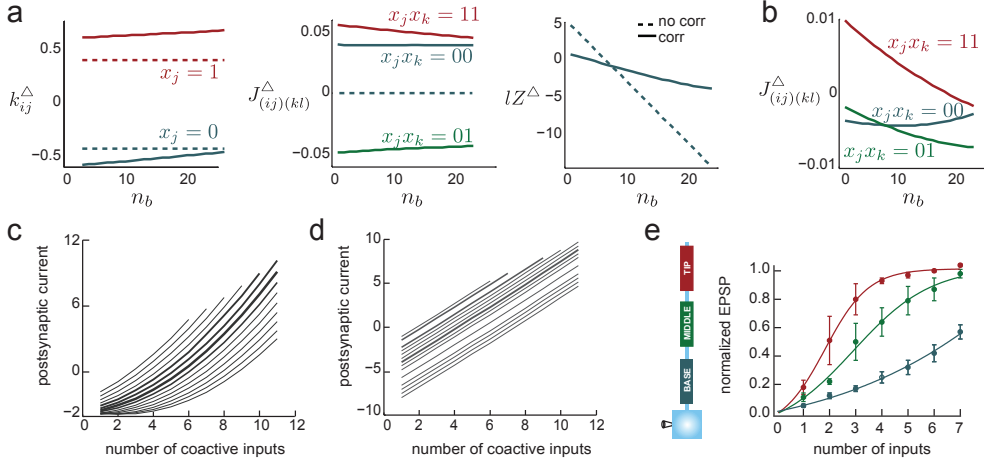

Figure 3: **Implications for neural dynamics. a.** R1: parameters for $I_i^{\text{rec}}$; linear modulation by network activity, $n_b$. **b.** R2: nonlinear modulation of pairwise term by network activity (cf. middle panel in a); other parameters have linear dependences on $n_b$. **c.** R1: Total current as function of number of coactivated inputs, $\sum_j W_{ij} x_j$; lines: different levels of neural excitability $\sum_j W_{ij}$, line widths scale with frequency of occurrence in a sample run. **d.** Same for R2. **e.** Nonlinear integration in dendrites, reproduced from [11], cf. curves in c.

Exact retrieval dynamics based on Eq. 10, but not respecting locality constraints, work substantially better in the presence of synaptic correlations, for all rules (Fig. 2d, yellow bars). It is important to note that for the XOR rule, which was supposed to be the closest analog to the covariance rule and hence afford simple recall dynamics [22], error rates stay above control, suggesting that it is actually a case in which even dependencies beyond 2nd-order correlation would need to be considered.

As in the additive case, exact recall dynamics are biologically implausible, as the total current to the neuron depends on the full weight matrix. It is possible to approximate the dynamics using strictly local information by replacing the nonlocal term by its mean, which, however, is no longer a constant, but rather a linear function of the total activity in the network, $n_b = \sum_{j \neq i} x_j$ [16]. Under this approximation, the current from recurrent connections corresponding to the evidence from the weights becomes:

$$I_i^{\text{rec}} = \log\left(\frac{P(\mathbf{W}|\mathbf{x}^{(1)})}{P(\mathbf{W}|\mathbf{x}^{(0)})}\right) = \sum_j k_{ij}^{\triangle}(\mathbf{x}) W_{ij} + \frac{1}{2}\sum_{jk} J_{(ij)(ik)}^{\triangle}(\mathbf{x}) W_{ij} W_{ik} - Z^{\triangle} \qquad (11)$$

where $i$ is the index of the neuron to be updated, and $\mathbf{x}^{(0/1)}$ activity vector has the to-be-updated neuron's activity set to 1 or 0, respectively, and all other components given by the current network state. The functions $k_{ij}^{\triangle}(\mathbf{x}) = k_{ij}(\mathbf{x}^{(1)}) - k_{ij}(\mathbf{x}^{(0)})$, $J_{(ij)(kl)}^{\triangle}(\mathbf{x}) = J_{(ij)(kl)}\left(\mathbf{x}^{(1)}\right) - J_{(ij)(kl)}\left(\mathbf{x}^{(0)}\right)$, and $Z^{\triangle} = \log\left(Z\left(\mathbf{x}^{(1)}\right)\right) - \log\left(Z\left(\mathbf{x}^{(0)}\right)\right)$ depend on the local activity at the indexed synapses, modulated by the number of active neurons in the network, $n_b$. This approximation is again consistent with our previous analysis, i.e. in the absence of synaptic correlations, the complex dynamics recover the simple case presented before. Importantly, this approximation also does about as well as exact dynamics (Fig. 2d, red bars).

For post-synaptically gated learning, comparing the parameters of the dynamics in the case of independent versus correlated synapses (Fig. 3a) reveals a modest modulation of the recurrent input by the total activity. More importantly, the net current to the postsynaptic neuron depends non-linearly (formally, quadratically) on the number of co-active inputs, $n_{W1} = \sum_j x_j W_{ij}$, (Fig. 3c), which is reminiscent of experimentally observed dendritic non-linearities [11] (Fig. 3e). Conversely, for the presynaptically gated learning rule, approximately optimal dynamics predict a non-monotonic modulation of activity by lateral inhibition (Fig. 3b), but linear neural integration (Fig. 3d).[8] Lastly, retrieval based on the XOR rule has the same form as the simple dynamics derived for the factorized case [16]. However, the total current has to be rescaled to compensate for the correlations introduced by reciprocal connections.

| | RULE | EXACT DYNAMICS | NEURAL IMPLEMENTATION |
|---|---|---|---|
| additive | covariance | strictly local, linear | linear feedback inh., homeostasis |
| | simple Hebbian | strictly local, nonlinear | nonlinear feedback inh. |
| | generalized Hebbian | semi-local, nonlinear | nonlinear feedback inh. |
| cascade | presyn. gated | nonlocal, nonlinear | nonlinear feedback inh., linear dendritic integr. |
| | postsyn. gated | nonlocal, nonlinear | linear feedback inh., non-linear dendritic integr. |
| | XOR | beyond correlations | ? |

Table 1: Results summary: circuit adaptations against correlations for different learning rules.

## 5    Discussion

Statistical dependencies between synaptic efficacies are a natural consequence of activity dependent synaptic plasticity, and yet their implications for network function have been unexplored. Here, in the context of an auto-associative memory network, we investigated the patterns of synaptic correlations induced by several well-known learning rules and their consequent effects on retrieval. We showed that most rules considered do indeed induce synaptic correlations and that failing to take them into account greatly damages recall. One fortuitous exception is the covariance rule, for which there are no synaptic correlations. This might explain why the bulk of classical treatments of auto-associative memories, using the covariance rule, could achieve satisfying capacity levels despite overlooking the issue of synaptic correlations [5, 24, 25].

In general, taking correlations into account optimally during recall requires dynamics in which there are non-local interactions between neurons. However, we derived approximations that perform well and are biologically realisable without such non-locality (Table 1). Examples include the modulation of neural responses by the total activity of the population, which could be mediated by feedback inhibition, and specific dendritic nonlinearities. In particular, for the post-synaptically gated learning rule, which may be viewed as an abstract model of hippocampal NMDA receptor-dependent plasticity, our model predicts a form of non-linear mapping of recurrent inputs into postsynaptic currents which is similar to experimentally observed dendritic integration in cortical pyramidal cells [11]. In general, the tight coupling between the synaptic plasticity used for encoding (manifested in patterns of synaptic correlations) and circuit dynamics offers an important route for experimental validation [2].

None of the rules governing synaptic plasticity that we considered perfectly reproduced the pattern of correlations in [6]; and indeed, exactly which rule applies in what region of the brain under which neuromodulatory influences is unclear. Furthermore, results in [6] concern the neocortex rather than the hippocampus, which is a more common target for models of auto-associative memory. Nonetheless, our analysis has shown that synaptic correlations matter for a range of very different learning rules that span the spectrum of empirical observations.

Another strategy to handle the negative effects of synaptic correlations is to weaken or eliminate them. For instance, in the palimpsest synaptic model [14], the deeper the cascade, the weaker the correlations, and so metaplasticity may have the beneficial effect of making recall easier. Another, popular, idea is to use very sparse patterns [21], although this reduces the information content of each one. More speculatively, one might imagine a process of off-line synaptic pruning or recoding, in which strong correlations are removed or the weights adjusted so that simple recall methods will work.

Here, we focused on second-order correlations. However, for plasticity rules such as XOR, we showed that this does not suffice. Rather, higher-order correlations would need to be considered, and thus, presumably higher-order interactions between neurons approximated. Finally, we know from work on neural coding of sensory stimuli that there are regimes in which correlations either help or hurt the informational quality of the code, assuming that decoding takes them into account. Given our results, it becomes important to look at the relative quality of different plasticity rules, assuming realizable decoding – it is not clear whether rules that strive to eliminate correlations will be bested by ones that do not.

**Acknowledgments** This work was supported by the Wellcome Trust (CS, ML), the Gatsby Charitable Foundation (PD), and the European Union Seventh Framework Programme (FP7/2007–2013) under grant agreement no. 269921 (BrainScaleS) (ML).

## Footnotes

[1]Complete connectivity simplifies the computation of the parameters for the optimal dynamics for cascade-like learning rules considered in the following, but is not necessary for the theory.

[2]For notational convenience, we use a column-vector form of the matrix of weight changes $\boldsymbol{\Omega}$, and weight matrix $\mathbf{W}$, marked by subscript $_|$.

[3]For additive learning rules, the current to neuron $i$ always depends only on synapses local to a neuron, but these can also include outgoing synapses of which the weight, $W_{\cdot i}$, should not influence its dynamics. We refer to such dynamics as 'semi-local'. For other learning rules, the optimal current to neuron $i$ may depend on all connections in the network, including $W_{jk}$ with $j, k \neq i$ ('non-local' dynamics).

[4]Other models, e.g. serial [19], could be used as well without qualitatively affecting the results.

[5]One could argue that this is the most biologically relevant as plasticity is often NMDA-receptor dependent, and hence it requires postsynaptic depolarisation for any effect to occur.

[6]This is just a generalisation of the simple dynamics which assume a first order max entropy model; moreover, the resulting weight distribution is a binary analog of the multivariate normal used in the additive case, allowing the two to be directly compared.

[7]Here, we ask whether it is possible to accommodate correlations in appropriate neural dynamics at all, ignoring the issue of how the optimal values for the parameters of the network dynamics would come about.

[8]The difference between the two rules emerges exclusively because of the constraint of strict locality of the approximation, since the exact form of the dynamics is essentially the same for the two.

# References

1. Sommer, F.T. & Dayan, P. Bayesian retrieval in associative memories with storage errors. *IEEE transactions on neural networks* **9**, 705–713 (1998).

2. Lengyel, M., Kwag, J., Paulsen, O. & Dayan, P. Matching storage and recall: hippocampal spike timing-dependent plasticity and phase response curves. *Nature Neuroscience* **8**, 1677–1683 (2005).

3. Lengyel, M. & Dayan, P. Uncertainty, phase and oscillatory hippocampal recall. *Advances in Neural Information Processing* (2007).

4. Savin, C., Dayan, P. & Lengyel, M. Two is better than one: distinct roles for familiarity and recollection in retrieving palimpsest memories. in *Advances in Neural Information Processing Systems, 24* (MIT Press, Cambridge, MA, 2011).

5. Hopfield, J.J. Neural networks and physical systems with emergent collective computational abilities. *Proc. Natl. Acad. Sci. USA* **76**, 2554–2558 (1982).

6. Song, S., Sjöström, P.J., Reigl, M., Nelson, S. & Chklovskii, D.B. Highly nonrandom features of synaptic connectivity in local cortical circuits. *PLoS biology* **3**, e68 (2005).

7. Dayan, P. & Abbott, L. *Theoretical Neuroscience* (MIT Press, 2001).

8. Averbeck, B.B., Latham, P.E. & Pouget, A. Neural correlations, population coding and computation. *Nature Reviews Neuroscience* **7**, 358–366 (2006).

9. Pillow, J.W. *et al.* Spatio-temporal correlations and visual signalling in a complete neuronal population. *Nature* **454**, 995–999 (2008).

10. Latham, P.E. & Nirenberg, S. Synergy, redundancy, and independence in population codes, revisited. *Journal of Neuroscience* **25**, 5195–5206 (2005).

11. Branco, T. & Häusser, M. Synaptic integration gradients in single cortical pyramidal cell dendrites. *Neuron* **69**, 885–892 (2011).

12. Hasselmo, M.E. & Bower, J.M. Acetylcholine and memory. *Trends Neurosci.* **16**, 218–222 (1993).

13. MacKay, D.J.C. Maximum entropy connections: neural networks. in *Maximum Entropy and Bayesian Methods, Laramie, 1990* (eds. Grandy, Jr, W.T. & Schick, L.H.) 237–244 (Kluwer, Dordrecht, The Netherlands, 1991).

14. Fusi, S., Drew, P.J. & Abbott, L.F. Cascade models of synaptically stored memories. *Neuron* **45**, 599–611 (2005).

15. Abraham, W.C. Metaplasticity: tuning synapses and networks for plasticity. *Nature Reviews Neuroscience* **9**, 387 (2008).

16. For details, see Supplementary Information.

17. Zhang, W. & Linden, D. The other side of the engram: experience-driven changes in neuronal intrinsic excitability. *Nature Reviews Neuroscience* (2003).

18. Engel, A., Englisch, H. & Schütte, A. Improved retrieval in neural networks with external fields. *Europhysics Letters (EPL)* **8**, 393–397 (1989).

19. Leibold, C. & Kempter, R. Sparseness constrains the prolongation of memory lifetime via synaptic metaplasticity. *Cerebral cortex (New York, N.Y. : 1991)* **18**, 67–77 (2008).

20. Amit, Y. & Huang, Y. Precise capacity analysis in binary networks with multiple coding level inputs. *Neural computation* **22**, 660–688 (2010).

21. Huang, Y. & Amit, Y. Capacity analysis in multi-state synaptic models: a retrieval probability perspective. *Journal of computational neuroscience* (2011).

22. Dayan Rubin, B. & Fusi, S. Long memory lifetimes require complex synapses and limited sparseness. *Frontiers in Computational Neuroscience* (2007).

23. Thouless, D.J., Anderson, P.W. & Palmer, R.G. Solution of 'Solvable model of a spin glass'. *Philosophical Magazine* **35**, 593–601 (1977).

24. Amit, D., Gutfreund, H. & Sompolinsky, H. Storing infinite numbers of patterns in a spin-glass model of neural networks. *Phys Rev Lett* **55**, 1530–1533 (1985).

25. Treves, A. & Rolls, E.T. What determines the capacity of autoassociative memories in the brain? *Network* **2**, 371–397 (1991).

